# Coarse sample complexity bounds for active learning

**Sanjoy Dasgupta**
UC San Diego
dasgupta@cs.ucsd.edu

## Abstract

We characterize the sample complexity of active learning problems in terms of a parameter which takes into account the distribution over the input space, the specific target hypothesis, and the desired accuracy.

## 1 Introduction

The goal of active learning is to learn a classifier in a setting where data comes unlabeled, and any labels must be explicitly requested and paid for. The hope is that an accurate classifier can be found by buying just a few labels.

So far the most encouraging theoretical results in this field are [7, 6], which show that if the hypothesis class is that of homogeneous (i.e. through the origin) linear separators, and the data is distributed uniformly over the unit sphere in $\mathbb{R}^d$, and the labels correspond perfectly to one of the hypotheses (i.e. the separable case) then at most $O(d \log d/\epsilon)$ labels are needed to learn a classifier with error less than $\epsilon$. This is exponentially smaller than the usual $\Omega(d/\epsilon)$ sample complexity of learning linear classifiers in a supervised setting.

However, generalizing this result is non-trivial. For instance, if the hypothesis class is expanded to include non-homogeneous linear separators, then even in just two dimensions, under the same benign input distribution, we will see that there are some target hypotheses for which active learning does not help much, for which $\Omega(1/\epsilon)$ labels are needed. In fact, in this example the label complexity of active learning depends heavily on the specific target hypothesis, and ranges from $O(\log 1/\epsilon)$ to $\Omega(1/\epsilon)$.

In this paper, we consider arbitrary hypothesis classes $\mathcal{H}$ of VC dimension $d < \infty$, and learning problems which are separable. We characterize the sample complexity of active learning in terms of a parameter which takes into account: (1) the distribution $\mathbb{P}$ over the input space $\mathcal{X}$; (2) the specific target hypothesis $h^* \in \mathcal{H}$; and (3) the desired accuracy $\epsilon$.

Specifically, we notice that distribution $\mathbb{P}$ induces a natural topology on $\mathcal{H}$, and we define a *splitting index* $\rho$ which captures the relevant local geometry of $\mathcal{H}$ in the vicinity of $h^*$, at scale $\epsilon$. We show that this quantity fairly tightly describes the sample complexity of active learning: any active learning scheme requires $\Omega(1/\rho)$ labels and there is a generic active learner which always uses at most $\tilde{O}(d/\rho)$ labels[1].

This $\rho$ is always at least $\epsilon$; if it is $\epsilon$ we just get the usual sample complexity of supervised

learning. But sometimes $\rho$ is a constant, and in such instances active learning gives an exponential improvement in the number of labels needed.

We look at various hypothesis classes and derive splitting indices for target hypotheses at different levels of accuracy. For homogeneous linear separators and the uniform input distribution, we easily find $\rho$ to be a constant – perhaps the most direct proof yet of the efficacy of active learning in this case. Most proofs have been omitted for want of space; the full details, along with more examples, can be found at [5].

## 2 Sample complexity bounds

### 2.1 Motivating examples

**Linear separators in $\mathbb{R}^1$**

Our first example is taken from [3, 4]. Suppose the data lie on the real line, and the classifiers are simple thresholding functions, $\mathcal{H} = \{h_w : w \in \mathbb{R}\}$:

$$h_w(x) = \begin{cases} 1 & \text{if } x \geq w \\ 0 & \text{if } x < w \end{cases}$$

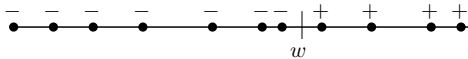

VC theory tells us that if the underlying distribution $\mathbb{P}$ is separable (can be classified perfectly by some hypothesis in $\mathcal{H}$), then in order to achieve an error rate less than $\epsilon$, it is enough to draw $m = O(1/\epsilon)$ random labeled examples from $\mathbb{P}$, and to return any classifier consistent with them. But suppose we instead draw $m$ *unlabeled* samples from $\mathbb{P}$. If we lay these points down on the line, their hidden labels are a sequence of 0's followed by a sequence of 1's, and the goal is to discover the point $w$ at which the transition occurs. This can be done with a binary search which asks for just $\log m = O(\log 1/\epsilon)$ labels. Thus, in this case active learning gives an *exponential* improvement in the number of labels needed.

Can we always achieve a label complexity proportional to $\log 1/\epsilon$ rather than $1/\epsilon$? A natural next step is to consider linear separators in *two* dimensions.

**Linear separators in $\mathbb{R}^2$**

Let $\mathcal{H}$ be the hypothesis class of linear separators in $\mathbb{R}^2$, and suppose the input distribution $\mathbb{P}$ is some density supported on the perimeter of the unit circle. It turns out that the positive results of the one-dimensional case do not generalize: there are some target hypotheses in $\mathcal{H}$ for which $\Omega(1/\epsilon)$ labels are needed to find a classifier with error rate less than $\epsilon$, no matter what active learning scheme is used.

To see this, consider the following possible target hypotheses (Figure 1, left): $h_0$, for which all points are positive; and $h_i$ ($1 \leq i \leq 1/\epsilon$), for which all points are positive except for a small slice $B_i$ of probability mass $\epsilon$.

The slices $B_i$ are explicitly chosen to be disjoint, with the result that $\Omega(1/\epsilon)$ labels are needed to distinguish between these hypotheses. For instance, suppose nature chooses a target hypothesis at random from among the $h_i, 1 \leq i \leq 1/\epsilon$. Then, to identify this target with probability at least $1/2$, it is necessary to query points in at least (about) half the $B_i$'s.

Thus for these particular target hypotheses, active learning offers no improvement in sample complexity. What about other target hypotheses in $\mathcal{H}$, for instance those in which the positive and negative regions are most evenly balanced? Consider the following active learning scheme:

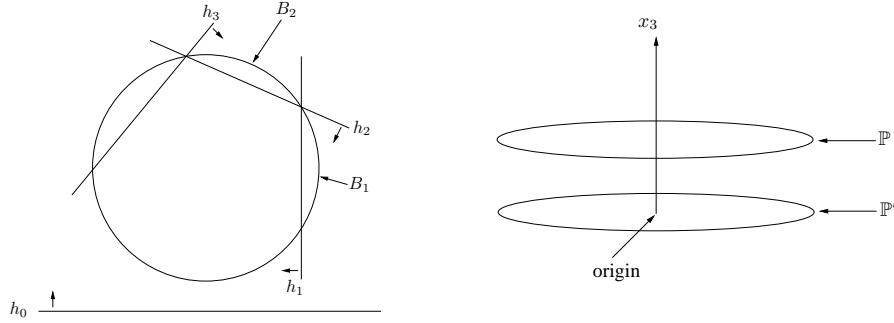

Figure 1: *Left:* The data lie on the circumference of a circle. Each $B_i$ is an arc of probability mass $\epsilon$. *Right:* The same distribution $\mathbb{P}$, lifted to 3-d, and with trace amounts of another distribution $\mathbb{P}'$ mixed in.

1. Draw a pool of $O(1/\epsilon)$ unlabeled points.

2. From this pool, choose query points at random until at least one positive and one negative point have been found. (If all points have been queried, then halt.)

3. Apply binary search to find the two boundaries between positive and negative on the perimeter of the circle.

For any $h \in \mathcal{H}$, define $i(h) = \min\{\text{positive mass of } h, \text{negative mass of } h\}$. It is not hard to see that when the target hypothesis is $h$, step (2) asks for $O(1/i(h))$ labels (with probability at least $9/10$, say) and step (3) asks for $O(\log 1/\epsilon)$ labels.

Thus even within this simple hypothesis class, the label complexity of active learning can run anywhere from $O(\log 1/\epsilon)$ to $\Omega(1/\epsilon)$, depending on the specific target hypothesis.

**Linear separators in $\mathbb{R}^3$**

In our two previous examples, the amount of unlabeled data needed was $O(1/\epsilon)$, exactly the usual sample complexity of supervised learning. We next turn to a case in which it is helpful to have significantly more unlabeled data than this.

Consider the distribution of the previous 2-d example: for concreteness, fix $\mathbb{P}$ to be uniform over the unit circle in $\mathbb{R}^2$. Now lift it into three dimensions by adding to each point $x = (x_1, x_2)$ a third coordinate $x_3 = 1$. Let $\mathcal{H}$ consist of *homogeneous* linear separators in $\mathbb{R}^3$. Clearly the bad cases of the previous example persist.

Suppose, now, that a trace amount $\tau$ of a second distribution $\mathbb{P}'$ is mixed in with $\mathbb{P}$ (Figure 1, right), where $\mathbb{P}'$ is uniform on the circle $\{x_1^2 + x_2^2 = 1, x_3 = 0\}$. The "bad" linear separators in $\mathcal{H}$ cut off just a small portion of $\mathbb{P}$ but nonetheless divide $\mathbb{P}'$ perfectly in half. This permits a three-stage algorithm: (1) using binary search on points from $\mathbb{P}'$, approximately identify the two places at which the target hypothesis $h^*$ cuts $\mathbb{P}'$; (2) use this to identify a positive and negative point of $\mathbb{P}$ (look at the midpoints of the positive and negative intervals in $\mathbb{P}'$); (3) do binary search on points from $\mathbb{P}$. Steps (1) and (3) each use just $O(\log 1/\epsilon)$ labels.

This $O(\log 1/\epsilon)$ label complexity is made possible by the presence of $\mathbb{P}'$ and is only achievable if the amount of unlabeled data is $\Omega(1/\tau)$, which could potentially be enormous. With less unlabeled data, the usual $\Omega(1/\epsilon)$ label complexity applies.

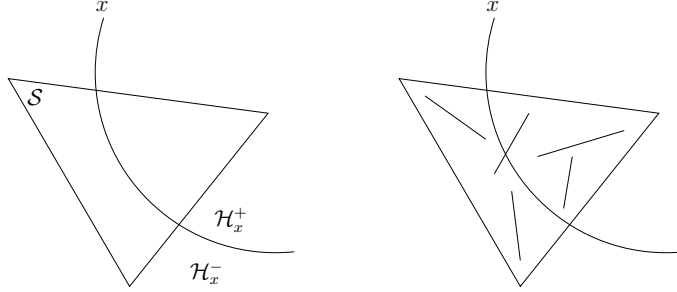

Figure 2: (a) $x$ is a cut through $\mathcal{H}$; (b) splitting edges.

## 2.2 Basic definitions

The sample complexity of supervised learning is commonly expressed as a function of the error rate $\epsilon$ and the underlying distribution $\mathbb{P}$. For active learning, the previous three examples demonstrate that it is also important to take into account the target hypothesis and the amount of unlabeled data. The main goal of this paper is to present one particular formalism by which this can be accomplished.

Let $\mathcal{X}$ be an instance space with underlying distribution $\mathbb{P}$. Let $\mathcal{H}$ be the hypothesis class, a set of functions from $\mathcal{X}$ to $\{0, 1\}$ whose VC dimension is $d < \infty$.

We are operating in a non-Bayesian setting, so we are not given a measure (prior) on the space $\mathcal{H}$. In the absence of a measure, there is no natural notion of the "volume" of the current version space. However, the distribution $\mathbb{P}$ does induce a natural distance function on $\mathcal{H}$, a pseudometric:
$$d(h, h') = \mathbb{P}\{x : h(x) \neq h'(x)\}.$$
We can likewise define the notion of neighborhood: $B(h, r) = \{h' \in \mathcal{H} : d(h, h') \leq r\}$.

We will be dealing with a *separable* learning scenario, in which all labels correspond perfectly to some concept $h^* \in \mathcal{H}$, and the goal is to find $h \in \mathcal{H}$ such that $d(h^*, h) \leq \epsilon$. To do this, it is sufficient to whittle down the version space to the point where it has diameter at most $\epsilon$, and to then return any of the remaining hypotheses. Likewise, if the diameter of the current version space is more than $\epsilon$ then any hypothesis chosen from it will have error more than $\epsilon/2$ with respect to the worst-case target. Thus, in a non-Bayesian setting, active learning is about *reducing the diameter* of the version space.

If our current version space is $\mathcal{S} \subset \mathcal{H}$, how can we quantify the amount by which a point $x \in \mathcal{X}$ reduces its diameter? Let $\mathcal{H}_x^+$ denote the classifiers that assign $x$ a value of 1, $\mathcal{H}_x^+ = \{h \in \mathcal{H} : h(x) = 1\}$, and let $\mathcal{H}_x^-$ be the remainder, which assign it a value of 0. We can think of $x$ as a cut through hypothesis space; see Figure 2(a). In this example, $x$ is clearly helpful, but it doesn't reduce the diameter of $\mathcal{S}$. And we cannot say that it reduces the *average* distance between hypotheses, since again there is no measure on $\mathcal{H}$. What $x$ seems to be doing is to reduce the diameter in a certain "direction". Is there some notion in arbitrary metric spaces which captures this intuition?

Consider any finite $Q \subset \mathcal{H} \times \mathcal{H}$. We will think of an element $(h, h') \in Q$ as an *edge* between *vertices* $h$ and $h'$. For us, each such edge will represent a pair of hypotheses which need to be distinguished from one another: that is, they are relatively far apart, so there is no way to achieve our target accuracy if both of them remain in the version space. We would hope that for any finite set of edges $Q$, there are queries that will remove a substantial fraction of them.

To this end, a point $x \in \mathcal{X}$ is said to $\rho$-*split* $Q$ if its label is guaranteed to reduce the number

of edges by a fraction $\rho > 0$, that is, if:

$$\max\{|Q \cap (\mathcal{H}_x^+ \times \mathcal{H}_x^+)|,\ |Q \cap (\mathcal{H}_x^- \times \mathcal{H}_x^-)|\} \ \leq\ (1-\rho)|Q|.$$

For instance, in Figure 2(b), the edges are $3/5$-split by $x$.

If our target accuracy is $\epsilon$, we only really care about edges of length more than $\epsilon$. So define

$$Q_\epsilon = \{(h, h') \in Q : d(h, h') > \epsilon\}.$$

Finally, we say that a subset of hypotheses $\mathcal{S} \subset \mathcal{H}$ is $(\rho, \epsilon, \tau)$-*splittable* if for all finite edge-sets $Q \subset \mathcal{S} \times \mathcal{S}$,

$$\mathbb{P}\{x : x\ \rho\text{-splits } Q_\epsilon\} \geq \tau.$$

Paraphrasing, at least a $\tau$ fraction of the distribution $\mathbb{P}$ is useful for splitting $\mathcal{S}$.[2] This $\tau$ gives a sense of how many unlabeled samples are needed. If $\tau$ is miniscule, then there are good points to query, but these will emerge only in an enormous pool of unlabeled data. It will soon transpire that the parameters $\rho, \tau$ play roughly the following roles:

$$\#\text{ labels needed} \propto 1/\rho, \quad \#\text{ of unlabeled points needed} \propto 1/\tau$$

A first step towards understanding them is to establish a trivial lower bound on $\rho$.

**Lemma 1** *Pick any* $0 < \alpha, \epsilon < 1$, *and any set* $\mathcal{S}$. *Then* $\mathcal{S}$ *is* $((1-\alpha)\epsilon, \epsilon, \alpha\epsilon)$-*splittable.*

*Proof.* Pick any finite edge-set $Q \subset \mathcal{S} \times \mathcal{S}$. Let $Z$ denote the number of edges of $Q_\epsilon$ cut by a point $x$ chosen at random from $\mathbb{P}$. Since the edges have length at least $\epsilon$, this $x$ has at least an $\epsilon$ chance of cutting any of them, whereby $\mathbb{E}Z \geq \epsilon|Q_\epsilon|$. Now,

$$\epsilon|Q_\epsilon| \ \leq\ \mathbb{E}Z \ \leq\ \mathbb{P}(Z \geq (1-\alpha)\epsilon|Q_\epsilon|) \cdot |Q_\epsilon| \ +\ (1-\alpha)\epsilon|Q_\epsilon|,$$

which after rearrangement becomes $\mathbb{P}(Z \geq (1-\alpha)\epsilon|Q_\epsilon|) \geq \alpha\epsilon$, as claimed. ∎

Thus, $\rho$ is always $\Omega(\epsilon)$; but of course, we hope for a much larger value. We will now see that the splitting index roughly characterizes the sample complexity of active learning.

## 2.3   Lower bound

We start by showing that if some region of the hypothesis space has a low splitting index, then it must contain hypotheses which are not conducive to active learning.

**Theorem 2** *Fix a hypothesis space* $\mathcal{H}$ *and distribution* $\mathbb{P}$. *Suppose that for some* $\rho, \epsilon < 1$ *and* $\tau < 1/2$, $\mathcal{S} \subset \mathcal{H}$ *is not* $(\rho, \epsilon, \tau)$-*splittable. Then any active learner which achieves an accuracy of* $\epsilon$ *on all target hypotheses in* $\mathcal{S}$, *with confidence* $> 3/4$ *(over the random sampling of data), either needs* $\geq 1/\tau$ *unlabeled samples or* $\geq 1/\rho$ *labels.*

*Proof.* Let $Q_\epsilon$ be the set of edges of length $> \epsilon$ which defies splittability, with vertices $\mathcal{V} = \{h : (h, h') \in Q_\epsilon \text{ for some } h' \in \mathcal{H}\}$. We'll show that in order to distinguish between hypotheses in $\mathcal{V}$, either $1/\tau$ unlabeled samples or $1/\rho$ queries are needed.

So pick less than $1/\tau$ unlabeled samples. With probability at least $(1-\tau)^{1/\tau} \geq 1/4$, none of these points $\rho$-splits $Q_\epsilon$; put differently, each of these potential queries has a bad outcome ($+$ or $-$) in which at most $\rho|Q_\epsilon|$ edges are eliminated. In this case there must be a target hypothesis in $\mathcal{V}$ for which at least $1/\rho$ labels are required. ∎

In our examples, we will apply this lower bound through the following simple corollary.

Let $S_0$ be an $\epsilon_0$-cover of $\mathcal{H}$
for $t = 1, 2, \ldots, T = \lg 2/\epsilon$:
$\quad S_t = \text{split}(S_{t-1}, 1/2^t)$
return any $h \in S_T$

*function split$(S, \Delta)$*
$\quad$ Let $Q_0 = \{(h, h') \in S \times S : d(h, h') > \Delta\}$
$\quad$ Repeat for $t = 0, 1, 2, \ldots$:
$\qquad$ Draw $m$ unlabeled points $x_{t1}, \ldots, x_{tm}$
$\qquad$ Query the $x_{ti}$ which maximally splits $Q_t$
$\qquad$ Let $Q_{t+1}$ be the remaining edges
$\quad$ until $Q_{t+1} = \emptyset$
$\quad$ return remaining hypotheses in $S$

Figure 3: A generic active learner.

**Corollary 3** *Suppose that in some neighborhood $B(h_0, \Delta)$, there are hypotheses $h_1, \ldots, h_N$ such that: (1) $d(h_0, h_i) > \epsilon$ for all $i$; and (2) the "disagree sets" $\{x : h_0(x) \neq h_i(x)\}$ are disjoint for different $i$.*

*Then for any $\tau$ and any $\rho > 1/N$, the set $B(h_0, \Delta)$ is not $(\rho, \epsilon, \tau)$-splittable . Any active learning scheme which achieves an accuracy of $\epsilon$ on all of $B(h_0, \Delta)$ must use at least $N$ labels for some of the target hypotheses, no matter how much unlabeled data is available.*

In this case, the distance metric on $h_0, h_1, \ldots, h_N$ can accurately be depicted as a *star* with $h_0$ at the center and with spokes leading to each $h_i$. Each query only cuts off one spoke, so $N$ queries are needed.

## 2.4 Upper bound

We now show a loosely matching upper bound on sample complexity, via an algorithm (Figure 3) which repeatedly halves the diameter of the remaining version space. For some $\epsilon_0$ less than half the target error rate $\epsilon$, it starts with an $\epsilon_0$-cover of $\mathcal{H}$: a set of hypotheses $S_0 \subset \mathcal{H}$ such that any $h \in \mathcal{H}$ is within distance $\epsilon_0$ of $S_0$. It is well-known that it is possible to find such an $S_0$ of size $\leq 2(2e/\epsilon_0 \ln 2e/\epsilon_0)^d$ [9](Theorem 5). The $\epsilon_0$-cover serves as a surrogate for the hypothesis class – for instance, the final hypothesis is chosen from it.

The algorithm is hopelessly intractable and is meant only to demonstrate the following upper bound.

**Theorem 4** *Let the target hypothesis be some $h^* \in \mathcal{H}$. Pick any target accuracy $\epsilon > 0$ and confidence level $\delta > 0$. Suppose $B(h^*, 4\Delta)$ is $(\rho, \Delta, \tau)$-splittable for all $\Delta \geq \epsilon/2$. Then there is an appropriate choice of $\epsilon_0$ and $m$ for which, with probability at least $1 - \delta$, the algorithm will draw $\tilde{O}((1/\epsilon) + (d/\rho\tau))$ unlabeled points, make $\tilde{O}(d/\rho)$ queries, and return a hypothesis with error at most $\epsilon$.*

This theorem makes it possible to derive label complexity bounds which are fine-tuned to the specific target hypothesis. At the same time, it is extremely loose in that no attempt has been made to optimize logarithmic factors.

## 3 Examples

### 3.1 Simple boundaries on the line

Returning to our first example, let $\mathcal{X} = \mathbb{R}$ and $\mathcal{H} = \{h_w : w \in \mathbb{R}\}$, where each $h_w$ is a threshold function $h_w(x) = \mathbf{1}(x \geq w)$. Suppose $\mathbb{P}$ is the underlying distribution on $\mathcal{X}$; for simplicity we'll assume it's a density, although the discussion can easily be generalized.

The distance measure $\mathbb{P}$ induces on $\mathcal{H}$ is

$$d(h_w, h_{w'}) \;=\; \mathbb{P}\{x : h_w(x) \neq h_{w'}(x)\} \;=\; \mathbb{P}\{x : w \leq x < w'\} \;=\; \mathbb{P}[w, w')$$

(assuming $w' \geq w$). Pick any accuracy $\epsilon > 0$ and consider any finite set of edges $Q = \{(h_{w_i}, h_{w_i'}) : i = 1, \ldots, n\}$, where without loss of generality the $w_i$ are in nondecreasing order, and where each edge has length greater than $\epsilon$: $\mathbb{P}[w_i, w_i') > \epsilon$. Pick $w$ so that $\mathbb{P}[w_{n/2}, w) = \epsilon$. It is easy to see that any $x \in [w_{n/2}, w)$ must eliminate at least half the edges in $Q$. Therefore, $\mathcal{H}$ is $(\rho = 1/2, \epsilon, \epsilon)$-splittable for any $\epsilon > 0$.

This echoes the simple fact that active-learning $\mathcal{H}$ is just a binary search.

## 3.2 Intervals on the line

The next case we consider is almost identical to our earlier example of 2-d linear separators (and the results carry over to that example, within constant factors). The hypotheses correspond to intervals on the real line: $\mathcal{X} = \mathbb{R}$ and $\mathcal{H} = \{h_{a,b} : a, b \in \mathbb{R}\}$, where $h_{a,b}(x) = \mathbf{1}(a \leq x \leq b)$. Once again assume $\mathbb{P}$ is a density. The distance measure it induces is $d(h_{a,b}, h_{a',b'}) \;=\; \mathbb{P}\{x : x \in [a,b] \cup [a',b'], x \notin [a,b] \cap [a',b']\} \;=\; \mathbb{P}([a,b]\Delta[a',b'])$, where $S \Delta T$ denotes symmetric difference $(S \cup T) \setminus (S \cap T)$.

Even in this very simple class, some hypotheses are much easier to active-learn than others.

*Hypotheses not amenable to active-learning.* Divide the real line into $1/\epsilon$ disjoint intervals, each with probability mass $\epsilon$, and let $\{h_i : i = 1, ..., 1/\epsilon\}$ denote the hypotheses taking value 1 on the corresponding intervals. Let $h_0$ be the everywhere-zero concept. Then these $h_i$ satisfy the conditions of Corollary 3; their star-shaped configuration forces a $\rho$-value of $\epsilon$, and active learning doesn't help at all in choosing amongst them.

*Hypotheses amenable to active learning.* The bad hypotheses are the ones whose intervals have small probability mass. We'll now see that larger concepts are not so bad; in particular, for any $h$ whose interval has mass $> 4\epsilon$, $B(h, 4\epsilon)$ is $(\rho = \Omega(1), \epsilon, \Omega(\epsilon))$-splittable.

Pick any $\epsilon > 0$ and any $h_{a,b}$ such that $\mathbb{P}[a,b] = r > 4\epsilon$. Consider a set of edges $Q$ whose endpoints are in $B(h_{a,b}, 4\epsilon)$ and which all have length $> \epsilon$. In the figure below, all lengths denote probability masses. Any concept in $B(h_{a,b}, 4\epsilon)$ (more precisely, its interval) must lie within the outer box and must contain the inner box (this inner box might be empty).

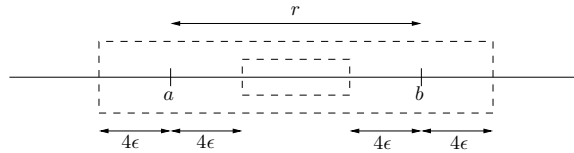

Any edge $(h_{a',b'}, h_{a'',b''}) \in Q$ has length $> \epsilon$, so $[a',b']\Delta[a'',b'']$ (either a single interval or a union of two intervals) has total length $> \epsilon$ and lies between the inner and outer boxes.

Now pick $x$ at random from the distribution $\mathbb{P}$ restricted to the space between the two boxes. This space has mass at most $16\epsilon$ and at least $4\epsilon$, of which at least $\epsilon$ is occupied by $[a',b']\Delta[a'',b'']$. Therefore $x$ separates $h_{a',b'}$ from $h_{a'',b''}$ with probability $\geq 1/16$.

Now let's look at all of $Q$. The expected number of edges split by our $x$ is at least $|Q|/16$, and therefore the probability that more than $|Q|/32$ edges are split is at least $1/32$. So $\mathbb{P}\{x : x \ (1/32)\text{-splits } Q\} \;\geq\; 4\epsilon/32 \;=\; \epsilon/8$.

To summarize, for any hypothesis $h_{a,b}$, let $i(h_{a,b}) = \mathbb{P}[a,b]$ denote the probability mass of its interval. Then for any $h \in \mathcal{H}$ and any $\epsilon < i(h)/4$, the set $B(h, 4\epsilon)$ is $(1/32, \epsilon, \epsilon/8)$-splittable. In short, once the version space is whittled down to $B(h, i(h)/4)$, efficient active

learning is possible. And the initial phase of getting to $B(h, i(h)/4)$ can be managed by random sampling, using $\tilde{O}(1/i(h))$ labels: not too bad when $i(h)$ is large.

### 3.3 Linear separators under the uniform distribution

The most encouraging positive result for active learning to date has been for learning homogeneous (through the origin) linear separators with data drawn uniformly from the surface of the unit sphere in $\mathbb{R}^d$. The splitting indices for this case [5] bring this out immediately:

**Theorem 5** *For any $h \in \mathcal{H}$, any $\epsilon \leq 1/(32\pi^2\sqrt{d})$, $B(h, 4\epsilon)$ is $(\frac{1}{8}, \epsilon, \Omega(\epsilon/\sqrt{d}))$-splittable.*

## 4 Related work and open problems

There has been a lot of work on a related model in which the points to be queried are synthetically constructed, rather than chosen from unlabeled data [1]. The expanded role of $\mathbb{P}$ in our model makes it substantially different, although a few intuitions do carry over – for instance, Corollary 3 generalizes the notion of *teaching dimension*[8].

We have already discussed [7, 4, 6]. One other technique which seems useful for active learning is to look at the unlabeled data and then place bets on certain target hypotheses, for instance the ones with large margin. This insight – nicely formulated in [2, 10] – is not specific to active learning and is orthogonal to the search issues considered in this paper.

In all the positive examples in this paper, a random data point which intersects the version space has a good chance of $\Omega(1)$-splitting it. This permits a naive active learning strategy, also suggested in [3]: just pick a random point whose label you are not yet sure of. On what kinds of problems will this work, and what are prototypical cases where more intelligent querying is needed?

**Acknowledgements.** I'm grateful to Yoav Freund for introducing me to this field; to Peter Bartlett, John Langford, Adam Kalai and Claire Monteleoni for helpful discussions; and to the anonymous NIPS reviewers for their detailed and perceptive comments.

## Footnotes

[1] The $\tilde{O}(\cdot)$ notation hides factors polylogarithmic in $d, 1/\epsilon, 1/\delta$, and $1/\tau$.

[2]Whenever an edge of length $l \geq \epsilon$ can be constructed in $S$, then by taking $Q$ to consist solely of this edge, we see that $\tau \leq l$. Thus we typically expect $\tau$ to be at most about $\epsilon$, although of course it might be a good deal smaller than this.

## References

[1] D. Angluin. Queries revisited. *ALT*, 2001.

[2] M.-F. Balcan and A. Blum. A PAC-style model for learning from labeled and unlabeled data. *Eighteenth Annual Conference on Learning Theory*, 2005.

[3] D. Cohn, L. Atlas, and R. Ladner. Improving generalization with active learning. *Machine Learning*, 15(2):201–221, 1994.

[4] S. Dasgupta. Analysis of a greedy active learning strategy. *NIPS*, 2004.

[5] S. Dasgupta. Full version of this paper at www.cs.ucsd.edu/˜dasgupta/papers/sample.ps.

[6] S. Dasgupta, A. Kalai, and C. Monteleoni. Analysis of perceptron-based active learning. *Eighteenth Annual Conference on Learning Theory*, 2005.

[7] Y. Freund, S. Seung, E. Shamir, and N. Tishby. Selective sampling using the query by committee algorithm. *Machine Learning Journal*, 28:133–168, 1997.

[8] S. Goldman and M. Kearns. On the complexity of teaching. *Journal of Computer and System Sciences*, 50(1):20–31, 1995.

[9] D. Haussler. Decision-theoretic generalizations of the PAC model for neural net and other learning applications. *Information and Computation*, 100(1):78–150, 1992.

[10] J. Shawe-Taylor, P. Bartlett, R. Williamson, and M. Anthony. Structural risk minimization over data-dependent hierarchies. *IEEE Transactions on Information Theory*, 44(5):1926–1940, 1998.